# Computing the Solution Path for the Regularized Support Vector Regression

**Lacey Gunter**
Department of Statistics
University of Michigan
Ann Arbor, MI 48109
lgunter@umich.edu

**Ji Zhu**[*]
Department of Statistics
University of Michigan
Ann Arbor, MI 48109
jizhu@umich.edu

## Abstract

In this paper we derive an algorithm that computes the entire solution path of the support vector regression, with essentially the same computational cost as fitting one SVR model. We also propose an unbiased estimate for the degrees of freedom of the SVR model, which allows convenient selection of the regularization parameter.

## 1  Introduction

The support vector regression (SVR) is a popular tool for function estimation problems, and it has been widely used on many real applications in the past decade, for example, time series prediction [1], signal processing [2] and neural decoding [3].

In this paper, we focus on the regularization parameter of the SVR, and propose an efficient algorithm that computes the entire regularized solution path; we also propose an unbiased estimate for the degrees of freedom of the SVR, which allows convenient selection of the regularization parameter.

Suppose we have a set of training data $(\boldsymbol{x}_1, y_1), \ldots, (\boldsymbol{x}_n, y_n)$, where the input $\boldsymbol{x}_i \in \mathbb{R}^p$ and the output $y_i \in \mathbb{R}$. Many researchers have noted that the formulation for the linear $\epsilon$-SVR can be written in a *loss + penalty* form [4]:

$$\min_{\beta_0, \boldsymbol{\beta}} \sum_{i=1}^{n} \left| y_i - \beta_0 - \boldsymbol{\beta}^{\mathsf{T}} \boldsymbol{x}_i \right|_{\epsilon} + \frac{\lambda}{2} \boldsymbol{\beta}^{\mathsf{T}} \boldsymbol{\beta} \tag{1}$$

where $|\xi|_{\epsilon}$ is the so called $\epsilon$-insensitive loss function:

$$|\xi|_{\epsilon} = \begin{cases} 0 & \text{if } |\xi| \leq \epsilon \\ |\xi| - \epsilon & \text{otherwise} \end{cases}$$

The idea is to disregard errors as long as they are less than $\epsilon$. Figure 1 plots the loss function. Notice that it has two non-differentiable points at $\pm\epsilon$. The regularization parameter $\lambda$ controls the trade-off between the $\epsilon$-insensitive loss and the complexity of the fitted model.

---

[*]To whom the correspondence should be addressed.

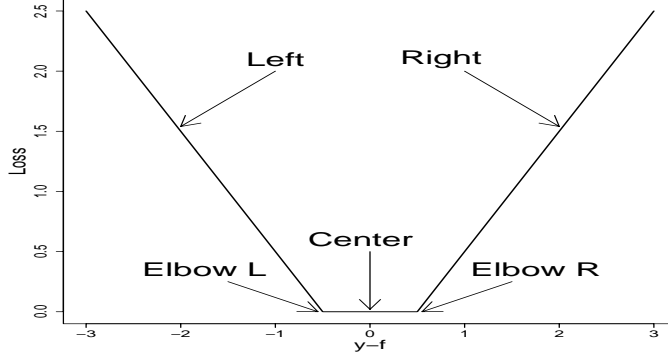

Figure 1: The $\epsilon$-insensitive loss function.

In practice, one often maps $\boldsymbol{x}$ into a high (often infinite) dimensional reproducing kernel Hilbert space (RKHS), and fits a nonlinear *kernel* SVR model [4]:

$$\min_{\beta_0,\boldsymbol{\theta}} \sum_{i=1}^{n} |y_i - f(\boldsymbol{x}_i)|_\epsilon + \frac{1}{2\lambda} \sum_{i=1}^{n} \sum_{i'=1}^{n} \theta_i \theta_{i'} K(\boldsymbol{x}_i, \boldsymbol{x}_{i'}) \qquad (2)$$

where $f(\boldsymbol{x}) = \beta_0 + \frac{1}{\lambda} \sum_{i=1}^{n} \theta_i K(\boldsymbol{x}, \boldsymbol{x}_i)$, and $K(\cdot, \cdot)$ is a positive-definite reproducing kernel that generates a RKHS. Notice that we write $f(\boldsymbol{x})$ in a way that involves $\lambda$ explicitly, and we will see later that $\theta_i \in [-1, 1]$.

Both (1) and (2) can be transformed into a quadratic programming problem, hence most commercially available packages can be used to solve the SVR. In the past years, many specific algorithms for the SVR have also been developed, for example, interior point algorithms [4-5], subset selection algorithms [6–7], and sequential minimal optimization [4, 8–9]. All these algorithms solve the SVR for a pre-fixed regularization parameter $\lambda$, and it is well known that an appropriate value of $\lambda$ is crucial for achieving small prediction error of the SVR.

In this paper, we show that the solution $\boldsymbol{\theta}(\lambda)$ is *piecewise linear* as a function of $\lambda$, which allows us to derive an efficient algorithm that computes the *exact entire solution path* $\{\boldsymbol{\theta}(\lambda), 0 \leq \lambda \leq \infty\}$. We acknowledge that this work was inspired by one of the authors' earlier work on the SVM setting [10].

Before delving into the technical details, we illustrate the concept of piecewise linearity of the solution path with a simple example. We generate 10 training observations using the famous $sinc(\cdot)$ function:

$$y = \frac{\sin(\pi x)}{\pi x} + e, \quad \text{where } x \sim U(-2\pi, 2\pi) \text{ and } e \sim N(0, 0.19^2)$$

We use the SVR with a 1-dimensional spline kernel

$$K(x, x') = 1 + k_1(x)k_1(x') + k_2(x)k_2(x') - k_4(|x - x'|) \qquad (3)$$

where $k_1(\cdot) = \cdot - 1/2, k_2 = (k_1^2 - 1/12)/2, k_4 = (k_1^4 - k_1^2/2 + 7/240)/24$. Figure 2 shows a subset of the piecewise linear solution path $\boldsymbol{\theta}(\lambda)$ as a function of $\lambda$.

In section 2, we describe the algorithm that computes the entire solution path of the SVR. In section 3, we propose an unbiased estimate for the degrees of freedom of the SVR, which can be used to select the regularization parameter $\lambda$. In section 4, we present numerical results on simulation data. We conclude the paper with a discussion section.

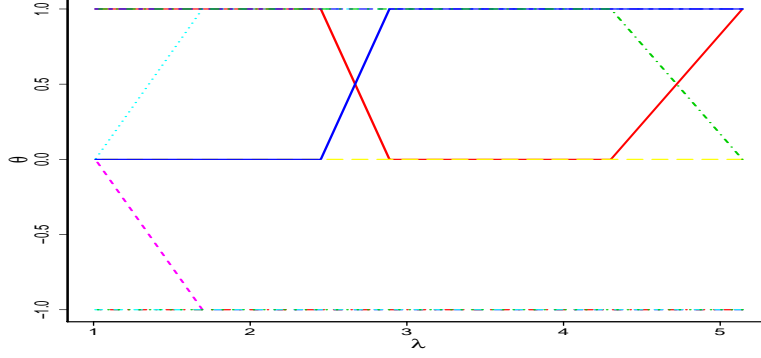

Figure 2: A subset of the solution path $\boldsymbol{\theta}(\lambda)$ as a function of $\lambda$.

## 2 Algorithm

For simplicity in notation, we describe the problem setup using the linear SVR, and the algorithm using the kernel SVR.

### 2.1 Problem Setup

The linear $\epsilon$-SVR (1) can be re-written in an equivalent way:

$$\min_{\beta_0, \boldsymbol{\beta}} \quad \sum_{i=1}^{n} (\xi_i + \delta_i) + \frac{\lambda}{2} \boldsymbol{\beta}^{\mathsf{T}} \boldsymbol{\beta}$$

$$\text{subject to} \quad -(\delta_i + \epsilon) \le y_i - f(\boldsymbol{x}_i) \le (\xi_i + \epsilon), \quad \xi_i, \delta_i \ge 0;$$

$$f(\boldsymbol{x}_i) = \beta_0 + \boldsymbol{\beta}^{\mathsf{T}} \boldsymbol{x}_i, \quad i = 1, \dots n$$

This gives us the Lagrangian primal function

$$L_P: \quad \sum_{i=1}^{n} (\xi_i + \delta_i) + \frac{\lambda}{2} \boldsymbol{\beta}^{\mathsf{T}} \boldsymbol{\beta} + \sum_{i=1}^{n} \alpha_i (y_i - f(\boldsymbol{x}_i) - \xi_i - \epsilon) -$$

$$\sum_{i=1}^{n} \gamma_i (y_i - f(\boldsymbol{x}_i) + \delta_i + \epsilon) - \sum_{i=1}^{n} \rho_i \xi_i - \sum_{i=1}^{n} \tau_i \delta_i.$$

Setting the derivatives to zero we arrive at:

$$\frac{\partial}{\partial \boldsymbol{\beta}}: \quad \boldsymbol{\beta} = \frac{1}{\lambda} \sum_{i=1}^{n} (\alpha_i - \gamma_i) \boldsymbol{x}_i \tag{4}$$

$$\frac{\partial}{\partial \beta_0}: \quad \sum_{i=1}^{n} \alpha_i = \sum_{i=1}^{n} \gamma_i \tag{5}$$

$$\frac{\partial}{\partial \xi_i}: \quad \alpha_i = 1 - \rho_i \tag{6}$$

$$\frac{\partial}{\partial \delta_i}: \quad \gamma_i = 1 - \tau_i \tag{7}$$

where the Karush-Kuhn-Tucker conditions are

$$\alpha_i (y_i - f(\boldsymbol{x}_i) - \xi_i - \epsilon) = 0 \tag{8}$$
$$\gamma_i (y_i - f(\boldsymbol{x}_i) + \delta_i + \epsilon) = 0 \tag{9}$$
$$\rho_i \xi_i = 0 \tag{10}$$
$$\tau_i \delta_i = 0 \tag{11}$$

Along with the constraint that our Lagrange multipliers must be non-negative, we can conclude from (6) and (7) that both $0 \le \alpha_i \le 1$ and $0 \le \gamma_i \le 1$. We also see from (8) and (9) that if $\alpha_i$ is positive, then $\gamma_i$ must be zero, and vice versa. These lead to the following relationships:

$$
\begin{array}{llllll}
y_i - f(\boldsymbol{x}_i) > \epsilon & \Rightarrow & \alpha_i = 1, & \xi_i > 0, & \gamma_i = 0, & \delta_i = 0; \\
y_i - f(\boldsymbol{x}_i) < -\epsilon & \Rightarrow & \alpha_i = 0, & \xi_i = 0, & \gamma_i = 1, & \delta_i > 0; \\
y_i - f(\boldsymbol{x}_i) \in (-\epsilon, \epsilon) & \Rightarrow & \alpha_i = 0, & \xi_i = 0, & \gamma_i = 0, & \delta_i = 0; \\
y_i - f(\boldsymbol{x}_i) = \epsilon & \Rightarrow & \alpha_i \in [0,1], & \xi_i = 0, & \gamma_i = 0, & \delta_i = 0; \\
y_i - f(\boldsymbol{x}_i) = -\epsilon & \Rightarrow & \alpha_i = 0, & \xi_i = 0, & \gamma_i \in [0,1], & \delta_i = 0.
\end{array}
$$

Using these relationships, we define the following sets that will be used later on when we are calculating the regularization path of the SVR:

- $\mathcal{R} = \{i : y_i - f(\boldsymbol{x}_i) > \epsilon,\ \alpha_i = 1, \gamma_i = 0\}$ (Right of the elbows)
- $\mathcal{E}_{\mathcal{R}} = \{i : y_i - f(\boldsymbol{x}_i) = \epsilon,\ 0 \le \alpha_i \le 1, \gamma_i = 0\}$ (Right elbow)
- $\mathcal{C} = \{i : -\epsilon < y_i - f(\boldsymbol{x}_i) < \epsilon,\ \alpha_i = 0, \gamma_i = 0\}$ (Center)
- $\mathcal{E}_{\mathcal{L}} = \{i : y_i - f(\boldsymbol{x}_i) = -\epsilon,\ \alpha_i = 0, 0 \le \gamma_i \le 0\}$ (Left elbow)
- $\mathcal{L} = \{i : y_i - f(\boldsymbol{x}_i) < -\epsilon,\ \alpha_i = 0, \gamma_i = 1\}$ (Left of the elbows)

Notice from (4) that for every $\lambda$, $\boldsymbol{\beta}$ is fully determined by the values of $\alpha_i$ and $\gamma_i$. For points in $\mathcal{R}$, $\mathcal{L}$ and $\mathcal{C}$, the values of $\alpha_i$ and $\gamma_i$ are known; therefore, the algorithm will focus on points resting at the two elbows $\mathcal{E}_{\mathcal{R}}$ and $\mathcal{E}_{\mathcal{L}}$.

## 2.2 Initialization

Initially, when $\lambda = \infty$ we can see from (4) that $\boldsymbol{\beta} = 0$. We can determine the value of $\beta_0$ via a simple 1-dimensional optimization. For lack of space, we focus on the case that all the values of $y_i$ are distinct, and furthermore, the initial sets $\mathcal{E}_{\mathcal{R}}$ and $\mathcal{E}_{\mathcal{L}}$ have at most one point combined (which is the usual situation). In this case $\beta_0$ will not be unique and each of the $\alpha_i$ and $\gamma_i$ will be either 0 or 1.

Since $\beta_0$ is not unique, we can focus on one particular solution path, for example, by always setting $\beta_0$ equal to one of its boundary values (thus keeping one point at an elbow). As $\lambda$ decreases, the range of $\beta_0$ shrinks toward zero and reaches zero when we have two points at the elbows, and the algorithm proceeds from there.

## 2.3 The Path

The formalized setup above can be easily modified to accommodate non-linear kernels; in fact, $\theta_i$ in (2) is equal to $\alpha_i - \gamma_i$. For the remaining portion of the algorithm we will use the kernel notation.

The algorithm focuses on the sets of points $\mathcal{E}_{\mathcal{R}}$ and $\mathcal{E}_{\mathcal{L}}$. These points have either $f(\boldsymbol{x}_i) = y_i - \epsilon$ with $\alpha_i \in [0,1]$, or $f(\boldsymbol{x}_i) = y_i + \epsilon$ with $\gamma_i \in [0,1]$. As we follow the path we will examine these sets until one or both of them change, at which point we will say an *event* has occurred. Thus events can be categorized as:

1. The initial event, for which two points must enter the elbow(s)
2. A point from $\mathcal{R}$ has just entered $\mathcal{E}_{\mathcal{R}}$, with $\alpha_i$ initially 1
3. A point from $\mathcal{L}$ has just entered $\mathcal{E}_{\mathcal{L}}$, with $\gamma_i$ initially 1
4. A point from $\mathcal{C}$ has just entered $\mathcal{E}_{\mathcal{R}}$, with $\alpha_i$ initially 0

5. A point from $\mathcal{C}$ has just entered $\mathcal{E}_\mathcal{L}$, with $\gamma_i$ initially 0

6. One or more points in $\mathcal{E}_\mathcal{R}$ and/or $\mathcal{E}_\mathcal{L}$ have just left the elbow(s) to join either $\mathcal{R}$, $\mathcal{L}$, or $\mathcal{C}$, with $\alpha_i$ and $\gamma_i$ initially 0 or 1

Until another event has occurred, all sets will remain the same. As a point passes through $\mathcal{E}_\mathcal{R}$ or $\mathcal{E}_\mathcal{L}$, its respective $\alpha_i$ or $\gamma_i$ must change from $0 \rightarrow 1$ or $1 \rightarrow 0$. Relying on the fact that $f(\boldsymbol{x}_i) = y_i - \epsilon$ or $f(\boldsymbol{x}_i) = y_i + \epsilon$ for all points in $\mathcal{E}_\mathcal{R}$ or $\mathcal{E}_\mathcal{L}$ respectively, we can calculate $\alpha_i$ and $\gamma_i$ for these points.

We use the subscript $\ell$ to index the sets above immediately after the $\ell$th event has occurred, and let $\alpha_i^\ell$, $\gamma_i^\ell$, $\beta_0^\ell$ and $\lambda^\ell$ be the parameter values immediately after the $\ell$th event. Also let $f^\ell$ be the function at this point. We define for convenience $\beta_{0,\lambda} = \lambda \cdot \beta_0$ and hence $\beta_{0,\lambda}^\ell = \lambda^\ell \cdot \beta_0^\ell$. Then since

$$f(\boldsymbol{x}) = \frac{1}{\lambda} \left( \sum_{i=1}^n (\alpha_i - \gamma_i) K(\boldsymbol{x}, \boldsymbol{x}_i) + \beta_{0,\lambda} \right)$$

for $\lambda^{\ell+1} < \lambda < \lambda^\ell$ we can write

$$
\begin{aligned}
f(\boldsymbol{x}) &= \left[ f(\boldsymbol{x}) - \frac{\lambda^\ell}{\lambda} f^\ell(\boldsymbol{x}) \right] + \frac{\lambda^\ell}{\lambda} f^\ell(\boldsymbol{x}) \\
&= \frac{1}{\lambda} \left[ \sum_{i \in \mathcal{E}_\mathcal{R}^\ell} \nu_i K(\boldsymbol{x}, \boldsymbol{x}_i) - \sum_{j \in \mathcal{E}_\mathcal{L}^\ell} \omega_j K(\boldsymbol{x}, \boldsymbol{x}_j) + \nu_0 + \lambda^\ell f^\ell(\boldsymbol{x}) \right],
\end{aligned}
$$

where $\nu_i = \alpha_i - \alpha_i^\ell$, $\omega_j = \gamma_j - \gamma_j^\ell$ and $\nu_0 = \beta_{0,\lambda} - \beta_{0,\lambda}^\ell$, and we can do the reduction in the second line since the $\alpha_i$ and $\gamma_i$ are fixed for all points in $\mathcal{R}^\ell$, $\mathcal{L}^\ell$, and $\mathcal{C}^\ell$ and all points remain in their respective sets. Suppose $|\mathcal{E}_\mathcal{R}^\ell| = n_R^\ell$ and $|\mathcal{E}_\mathcal{L}^\ell| = n_L^\ell$, so for the $n_R^\ell + n_L^\ell$ points staying at the elbows we have (after some algebra) that

$$\frac{1}{y_k - \epsilon} \left[ \sum_{i \in \mathcal{E}_\mathcal{R}^\ell} \nu_i K(\boldsymbol{x}_k, \boldsymbol{x}_i) - \sum_{j \in \mathcal{E}_\mathcal{L}^\ell} \omega_j K(\boldsymbol{x}_k, \boldsymbol{x}_j) + \nu_0 \right] = \lambda - \lambda^\ell, \quad \forall k \in \mathcal{E}_\mathcal{R}^\ell$$

$$\frac{1}{y_m + \epsilon} \left[ \sum_{i \in \mathcal{E}_\mathcal{R}^\ell} \nu_i K(\boldsymbol{x}_m, \boldsymbol{x}_i) - \sum_{j \in \mathcal{E}_\mathcal{L}^\ell} \omega_j K(\boldsymbol{x}_m, \boldsymbol{x}_j) + \nu_0 \right] = \lambda - \lambda^\ell, \quad \forall m \in \mathcal{E}_\mathcal{L}^\ell$$

Also, by condition (5) we have that

$$\sum_{i \in \mathcal{E}_\mathcal{R}^\ell} \nu_i - \sum_{j \in \mathcal{E}_\mathcal{L}^\ell} \omega_j = 0$$

This gives us $n_R^\ell + n_L^\ell + 1$ linear equations we can use to solve for each of the $n_R^\ell + n_L^\ell + 1$ unknown variables $\nu_i, \omega_j$ and $\nu_0$. Notice this system is linear in $\lambda - \lambda^\ell$, which implies that $\alpha_i$, $\gamma_j$ and $\beta_{0,\lambda}$ change linearly in $\lambda - \lambda^\ell$. So we can write:

$$\alpha_i = \alpha_i^\ell + (\lambda - \lambda^\ell) b_i \quad \forall i \in \mathcal{E}_\mathcal{R}^\ell \tag{12}$$

$$\gamma_j = \gamma_j^\ell + (\lambda - \lambda^\ell) b_j \quad \forall j \in \mathcal{E}_\mathcal{L}^\ell \tag{13}$$

$$\beta_{0,\lambda} = \beta_{0,\lambda}^\ell + (\lambda - \lambda^\ell) b_0 \tag{14}$$

$$f(\boldsymbol{x}) = \frac{\lambda^\ell}{\lambda} \left[ f^\ell(\boldsymbol{x}) - h^\ell(\boldsymbol{x}) \right] + h^\ell(\boldsymbol{x}) \tag{15}$$

where $(b_i, b_j, b_0)$ is the solution when $\lambda - \lambda^\ell$ is equal to 1, and

$$h^\ell(\boldsymbol{x}) = \sum_{i \in \mathcal{E}_\mathcal{R}^\ell} b_i K(\boldsymbol{x}, \boldsymbol{x}_i) - \sum_{j \in \mathcal{E}_\mathcal{L}^\ell} b_j K(\boldsymbol{x}, \boldsymbol{x}_j) + b_0.$$

Given $\lambda_\ell$, equations (12), (13) and (15) allow us to compute the $\lambda$ at which the next event will occur, $\lambda_{\ell+1}$. This will be the largest $\lambda$ less than $\lambda_\ell$, such that either $\alpha_i$ for $i \in \mathcal{E}_\mathcal{R}^\ell$ reaches 0 or 1, or $\gamma_j$ for $j \in \mathcal{E}_\mathcal{L}^\ell$ reaches 0 or 1, or one of the points in $\mathcal{R}$, $\mathcal{L}$ or $\mathcal{C}$ reaches an elbow.

We terminate the algorithm either when the sets $\mathcal{R}$ and $\mathcal{L}$ become empty, or when $\lambda$ has become sufficiently close to zero. In the later case we must have $f^\ell - h^\ell$ sufficiently small as well.

## 2.4 Computational cost

The major computational cost for updating the solutions at any event $\ell$ involves two things: solving the system of $(n_R^\ell + n_L^\ell)$ linear equations, and computing $h^\ell(\boldsymbol{x})$. The former takes $O((n_R^\ell + n_L^\ell)^2)$ calculations by using inverse updating and downdating since the elbow sets usually differ by only one point between consecutive events, and the latter requires $O(n(n_R^\ell + n_L^\ell))$ computations.

According to our experience, the total number of steps taken by the algorithm is on average some small multiple of $n$. Letting $m$ be the average size of $\mathcal{E}_\mathcal{R}^\ell \cup \mathcal{E}_\mathcal{L}^\ell$, then the approximate computational cost of the algorithm is $O\left(cn^2m + nm^2\right)$, which is comparable to a single SVR fitting algorithm that uses quadratic programming.

# 3 The Degrees of Freedom

The *degrees of freedom* is an informative measure of the complexity of a fitted model. In this section, we propose an unbiased estimate for the degrees of freedom of the SVR, which allows convenient selection of the regularization parameter $\lambda$.

Since the usual goal of regression analysis is to minimize the predicted squared-error loss, we study the degrees of freedom using Stein's unbiased risk estimation (SURE) theory [11]. Given $\boldsymbol{x}$, assuming $y$ is generated according to a homoskedastic model:

$$y \sim (\mu(\boldsymbol{x}), \sigma^2)$$

where $\mu$ is the true mean and $\sigma^2$ is the common variance. Then the degrees of freedom of a fitted model $f(\boldsymbol{x})$ can be defined as

$$\mathrm{df}(f) = \sum_{i=1}^n \mathrm{cov}(f(\boldsymbol{x}_i), y_i)/\sigma^2$$

Stein showed that under mild conditions, $\sum_{i=1}^n \partial f_i / \partial y_i$ is an unbiased estimate of $\mathrm{df}(f)$. It turns out that for the SVR model, for every fixed $\lambda$, $\sum_{i=1}^n \partial f_i / \partial y_i$ has an extremely simple formula:

$$\widehat{\mathrm{df}} \equiv \sum_{i=1}^n \frac{\partial f_i}{\partial y_i} = |\mathcal{E}_\mathcal{R}| + |\mathcal{E}_\mathcal{L}| \qquad (16)$$

Therefore, $|\mathcal{E}_\mathcal{R}| + |\mathcal{E}_\mathcal{L}|$ is a convenient unbiased estimate for the degrees of freedom of $f(\boldsymbol{x})$. Due to the space restriction, we omit the proof here, but make a note that the proof relies on our SVR algorithm.

In applying (16) to select the regularization parameter $\lambda$, we plug it into the GCV criterion [12] for model selection:

$$\frac{\sum_{i=1}^{n}(y_i - f(\boldsymbol{x}_i))^2}{(n - \widehat{\mathrm{df}})^2}$$

The advantages of this criterion are that it does not assume a known $\sigma^2$, and it avoids cross-validation, which is computationally intensive. In practice, we can first use our efficient algorithm to compute the entire solution path, then identify the appropriate value of $\lambda$ that minimizes the GCV criterion.

## 4   Numerical Results

To demonstrate our algorithm and the selection of $\lambda$ using the GCV criterion, we show numerical results on simulated data. We consider both additive and multiplicative kernels using the 1-dimensional spline kernel (3), which are respectively

$$K(\boldsymbol{x}, \boldsymbol{x}') = \sum_{j=1}^{p} K(x_j, x_j') \quad \text{and} \quad K(\boldsymbol{x}, \boldsymbol{x}') = \prod_{j=1}^{p} K(x_j, x_j')$$

Simulations were based on the following four functions [13]:

1. $f(x) = \frac{sin(\pi x)}{\pi x} + e_1, \quad x \in (-2\pi, 2\pi)$

2. $f(\boldsymbol{x}) = 0.1e^{4x_1} + \frac{1}{1+e^{-20(x_2 - .5)}} + 3x_3 + 2x_4 + x_5 + e_2, \quad \boldsymbol{x} \in (0,1)^2$

3. $f(R, \omega, L, C) = \left[R^2 + \left(\omega L + \frac{1}{\omega C}\right)^2\right]^{1/2} + e_3,$

4. $f(R, \omega, L, C) = \tan^{-1}\left[\frac{\omega L + \frac{1}{\omega C}}{R}\right] + e_4,$
   where $(R, \omega, L, C) \in (0, 100) \times (2\pi(20, 280)) \times (0, 1) \times (1, 11)$

$e_i$ are distributed as $N(0, \sigma_i^2)$, where $\sigma_1 = 0.19, \sigma_2 = 1, \sigma_3 = 218.5, \sigma_4 = 0.18$.

We generated 300 training observations from each function along with 10,000 validation observations and 10,000 test observations. For the first two simulations we used the additive 1-dimensional spline kernel and for the second two simulations the multiplicative 1-dimensional spline kernel. We then found the $\lambda$ that minimized the GCV criterion. The validation set was used to select the *gold standard* $\lambda$ which minimized the prediction MSE. Using these $\lambda$'s we calculated the prediction MSE with the test data for each criterion. After repeating this for 20 times, the average MSE and standard deviation for the MSE can be seen in Table 1, which indicates the GCV criterion performs closely to optimal.

Table 1: Simulation results of $\lambda$ selection for SVR

| $f(\boldsymbol{x})$ | MSE-Gold Standard | MSE-GCV |
|---|---|---|
| 1 | 0.0385 (0.0011) | 0.0389 (0.0011) |
| 2 | 1.0999 (0.0367) | 1.1120 (0.0382) |
| 3 | 50095 (1358) | 50982 (2205) |
| 4 | 0.0459 (0.0023) | 0.0471 (0.0028) |

# 5 Discussion

In this paper, we have proposed an efficient algorithm that computes the entire regularization path of the SVR. We have also proposed the GCV criterion for selecting the best $\lambda$ given the entire path. The GCV criterion seems to work sufficiently well on the simulation data. However, we acknowledge that according to our experience on real data sets (not shown here due to lack of the space), the GCV criterion sometimes tends to over-fit the model. We plan to explore this issue further.

Due to the difficulty of also selecting the best $\epsilon$ for the SVR, an alternate algorithm exists that automatically adjusts the value of $\epsilon$, called the $\nu$-SVR [4]. In this scenario, $\epsilon$ is treated as another free parameter. Using arguments similar to those for $\beta_0$ in our above algorithm, one can show that $\epsilon$ is piecewise linear in $1/\lambda$ and its path can be calculated similarly.

## Acknowledgments

We would like to thank Saharon Rosset for helpful comments. Gunter and Zhu are partially supported by grant DMS-0505432 from the National Science Foundation.

## References

[1] Müler K, Smola A, Rätsch G, Schölkopf B, Kohlmorgen J & Vapnik V (1997) Predicting time series with support vector machines. *Artificial Neural Networks*, 999-1004.

[2] Vapnik V, Golowich S & Smola A (1997) Support vector method for function approximation, regression estimation, and signal processing. *NIPS* **9**.

[3] Shpigelman L, Crammer K, Paz R, Vaadia E & Singer Y (2004) A temporal kernel-based model for tracking hand movements from neural activities. *NIPS* **17**, 1273-1280.

[4] Smola A & Schölkopf B (2004) A tutorial on support vector regression. *Statistics and Computing* **14**: 199-222.

[5] Vanderbei, R. (1994) LOQO: An interior point code for quadratic programming. *Technical Report SOR-94-15, Princeton University.*

[6] Osuna E, Freund R & Girosi F (1997) An improved training algorithm for support vector machines. *Neural Networks for Signal Processing*, 276-284.

[7] Joachims T (1999) Making large-scale SVM learning practical. *Advances in Kernel Methods – Support Vector Learning*, 169-184.

[8] Platt J (1999) Fast training of support vector machines using sequential minimal optimization. *Advances in Kernel Methods – Support Vector Learning*, 185-208.

[9] Keerthi S, Shevade S, Bhattacharyya C & Murthy K (1999) Improvements to Platt's SMO algorithm for SVM classifier design. *Technical Report CD-99-14, NUS.*

[10] Hastie, T., Rosset, S., Tibshirani, R. & Zhu, J. (2004) The Entire Regularization Path for the Support Vector Machine. *JMLR*, **5**, 1391-1415.

[11] Stein, C. (1981) Estimation of the mean of a multivariate normal distribution. *Annals of Statistics* **9**: 1135-1151.

[12] Craven, P. & Wahba, G. (1979) Smoothing noisy data with spline function. *Numerical Mathematics* 31: 377-403.

[13] Friedman, J. (1991) Multivariate Adaptive Regression Splines. *Annals of Statistics* **19**: 1-67.
